# Retinogeniculate Development: The Role of Competition and Correlated Retinal Activity

**Ron Keesing***
Dept. of Physiology
U.C. San Francisco
San Francisco, CA 94143
keesing@phy.ucsf.edu

**David G. Stork**
*Ricoh California Research Center
2882 Sand Hill Rd., Suite 115
Menlo Park, CA 94025
stork@crc.ricoh.com

**Carla J. Shatz**
Dept. of Neurobiology
Stanford University
Stanford, CA
94305

## Abstract

During visual development, projections from retinal ganglion cells (RGCs) to the lateral geniculate nucleus (LGN) in cat are refined to produce ocular dominance layering and precise topographic mapping. Normal development depends upon activity in RGCs, suggesting a key role for activity-dependent synaptic plasticity. Recent experiments on prenatal retina show that during early development, "waves" of activity pass across RGCs (Meister, et al., 1991). We provide the first simulations to demonstrate that such retinal waves, in conjunction with Hebbian synaptic competition and early arrival of contralateral axons, can account for observed patterns of retinogeniculate projections in normal and experimentally-treated animals.

## 1 INTRODUCTION

During the development of the mammalian visual system, initially diffuse axonal inputs are refined to produce the precise and orderly projections seen in the adult. In the lateral geniculate nucleus (LGN) of the cat, projections arriving from retinal ganglion cells (RGCs) of both eyes are initially intermixed, and they gradually segregate before birth to form alternating layers containing axons from only one eye. At the same time, the branching patterns of individual axons are refined, with increased growth in topographically correct locations. Axonal segregation and refinement depends upon

presynaptic activity — blocking such activity disrupts normal development (Sretavan, et al., 1988; Shatz & Stryker, 1988). These and findings in other vertebrates (Cline, et al., 1987) suggest that synaptic plasticity may be an essential factor in segregation and modification of RGC axons (Shatz, 1990).

Previous models of visual development based on synaptic plasticity (Miller, et al., 1989; Whitelaw & Cowan, 1981) required an assumption of spatial correlations in RGC activity for normal development. This assumption may have been justified for geniculo*cortical* development, since much of this occurs postnatally: visual stimulation provides the correlations. The assumption was more difficult to justify for retino*geniculate* development, since this occurs *prenatally* — before any optical stimulation.

The first strong evidence for correlated activity before birth has recently emerged in the retinogenculate system: wave-like patterns of synchronized activity pass across the prenatal retina, generating correlations between neighboring cells' activity (Meister, et al., 1991). We believe our model is the first to incorporate these important results.

We propose that during visual development, projections from both eyes compete to innervate LGN neurons. Contralateral projections, which reach the LGN earlier, may have a slight advantage in competing to innervate cells of the LGN located farthest from the optic tract. Retinal waves of activity could reinforce this segregation and improve the precision of topographic mapping by causing weight changes within the same eye — and particularly within the same region of the same eye — to be highly correlated. Unlike similar models of *cortical* development, our model does not require lateral interactions between post-synaptic cells — available evidence suggests that lateral inhibition is not present during early development (Shotwell, et al., 1986). Our model also incorporates axon *growth* — an essential aspect of retinogeniculate development, since the growth and branching of axons toward their ultimate targets occurs simultaneously with synaptic competition. Moreover, synaptic competition may provide cues for growth (Shatz & Stryker, 1988). We consider the possibility that diffusing intracellular signals indicating local synaptic strength guide axon growth.

Below we present simulations which show that this model can account for development in normal and experimentally-treated animals. We also predict the outcomes of novel experiments currently underway.

## 2   SIMULATIONS

Although the LGN is, of course, three-dimensional, in our model we represent just a single two-dimensional LGN slice, ten cells wide and eight cells high. The retina is then *one*-dimensional: 50 cells long in our simulations. (This ratio of widths, 50/10, is roughly that found in the developing cat.) In order to eliminate edge effects, we "wrap" the retina into a ring; likewise we wrap the LGN into a cylinder.

Development of projections to the LGN is modelled in the following way: projections from all fifty RGCs of the contralateral eye arrive at the base of the LGN before those of the ipsilateral eye. A very rough topographic map is imposed, corresponding to coarse topography which might be supplied by chemical gradients (Wolpert, 1978). Development is then modelled as a series of growth steps, each separated by a period of Hebb-style synaptic competition (Wigstrom & Gustafson, 1985). During competition, synapses are strengthened when pre- and post-synaptic activity are sufficiently correlated,

and they are weakened otherwise. More specifically, for a given RGC cell i with activity $a_i$, the strength of synapse $w_{ij}$ to LGN cell j is changed according to:

$$\Delta w_{ij} = \varepsilon \, (a_i - \alpha)(a_j - \beta) \tag{1}$$

where $\alpha$ and $\beta$ are threshholds and $\varepsilon$ a learning rate. If a "wave" of retinal activity is present, the activity of RGC cells is determined as a probability of firing based on a Gaussian function of the distance from the center of the wavefront. LGN cell activity is equal to the sum of weighted inputs from RGC cells.

After each wave, the total synaptic weight supported by each RGC cell i is renormalized linearly:

$$w_{ij}(t+1) = \frac{w_{ij}(t)}{\sum_k w_{ik}(t)} \tag{2}$$

The weights supported by each LGN cell are also renormalized, gradually driving them toward some target value T:

$$w_{ij}(t+1) = w_{ij}(t) + [T - \sum_k w_{kj}(t)] \tag{3}$$

Renormalization reflects the notion that there is a limited amount of synaptic weight which can be supported by any neuron.

During growth steps, connections are modified based on the strength of neighboring synapses from the same RGC cell. After normalization, connections grow or retract according to:

$$w_{ij}(t+1) = w_{ij}(t) + \gamma \sum_{neighbors} w_{ik}(t) \tag{4}$$

where $\gamma$ is a constant term. Equation 4 shows that weights in areas of high synaptic strength will increase more than those elsewhere.

# 3 RESULTS

Synaptic competition, in conjunction with waves of pre-synaptic activity and early arrival of contralateral axons, can account for pattens of growth seen in normal and experimentally-treated animals. In the presence of synaptic competition, modelled axons from each eye segregate to occupy discrete layers of the LGN — precisely what is seen in normal development. In the absence of competition, as in treatment with the drug TTX, axons arborize throughout the LGN (Figure 1).

The segregation and refinement of retinal inputs to the LGN is best illustrated by the formation of ocular dominance patterns and topographic ordering. In simulations of normal development, where retinal waves are combined with early arrival of contalateral inputs, strong ocular dominance layers are formed: LGN neurons farthest from the optic tract receive synaptic inputs solely from the contralateral eye and those closer receive only ipsilateral inputs (Figure2, Competition). The development of these ocular dominance patterns is gradual: early in development, a majority of LGN neurons receive inputs from *both* eyes. When synaptic competition is eliminated, there is no segregation into eye-specific layers — LGN neurons receive significant synaptic inputs from both eyes. These results are consistent with labelling studies of cat development (Shatz & Stryker, 1988).

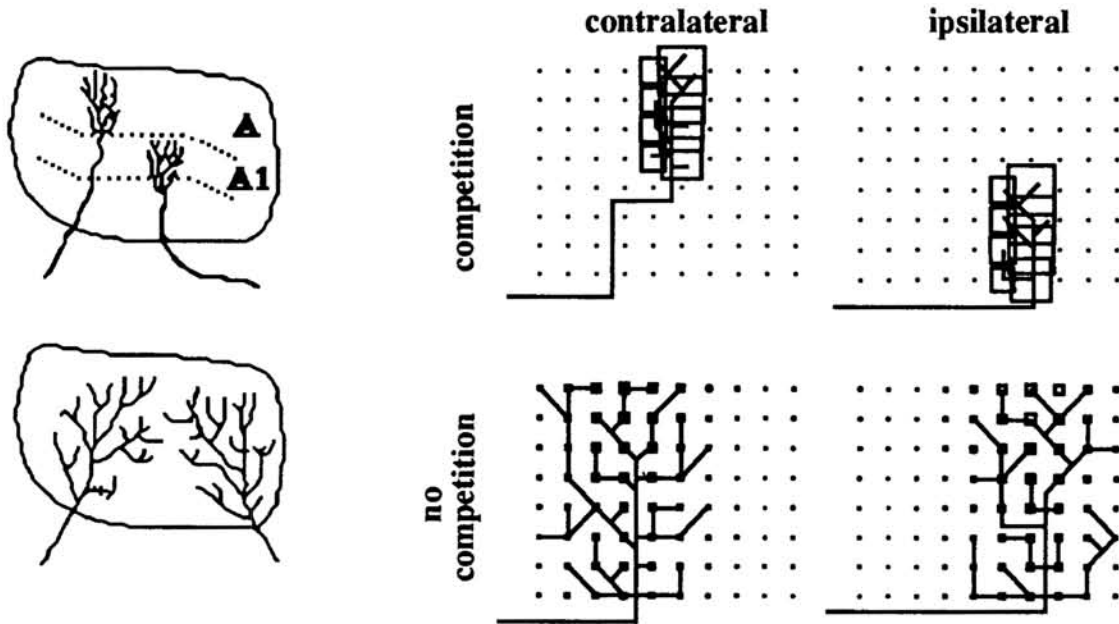

Figure 1: Retinogeniculate projections in vivo (adapted from Sretavan, et al., 1988.) (left), and simulation results (right). In the presence of competition (top), arbors are narrow and spatially localized, confined to the appropriate ocular dominance layer. In the absence of such competition (bottom), contralateral and ipsilateral projections are diffuse; there is no discernible ocular dominance pattern. During simulations, projections are represented by synapses throughout the LGN slice, shown as squares; the particular arborization patterns shown above are inferred from related simulations.

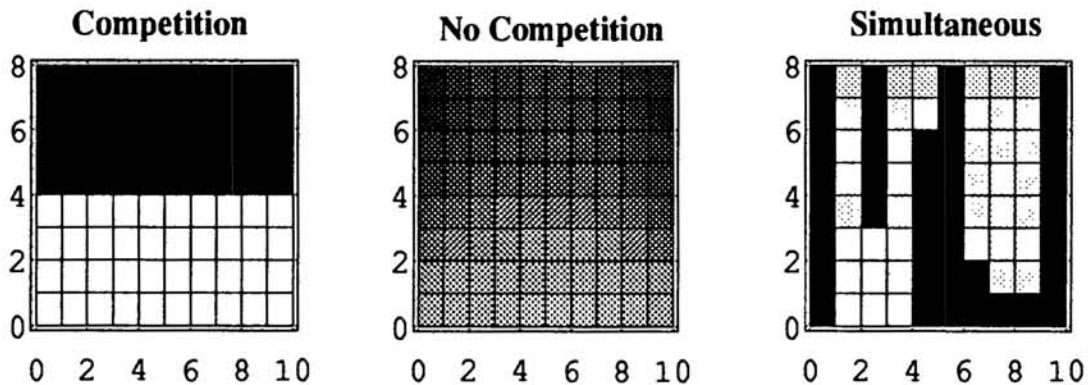

Figure 2: Ocular dominance at the end of development. Dark color indicates strongest synapses from the contralateral eye, light indicates strongest synapses from ipsilateral, and gray indicates significant synapses from both eyes. In the presence of competition, LGN cells segregate into eye-specific layers, with the contralateral eye dominating cells which are farthest from the optic tract (base). When competition is eliminated (No Competition), as in the addition of the drug TTX, there is no segregation into layers and LGN cells receive significant inputs from *both* eyes. These simulations reproduced results from cat development. When inputs from both retinae arrive simultaneously (Simultaneous), ocular dominance "patches" are established, similar to those observed in normal cortical development.

Retinal waves cause the activity of neighboring RGCs to be highly correlated. When combined with synaptic competition, these waves lead to a refinement of topographic ordering of retinogeniculate projections. During development, the coarse topography imposed as RGC axons enter the LGN is refined to produce an accurate, precise mapping of retinal inputs (Figure 3, Competition). Without competition, there is no refinement of topography, and the coarse initial mapping remains.

**Competition**

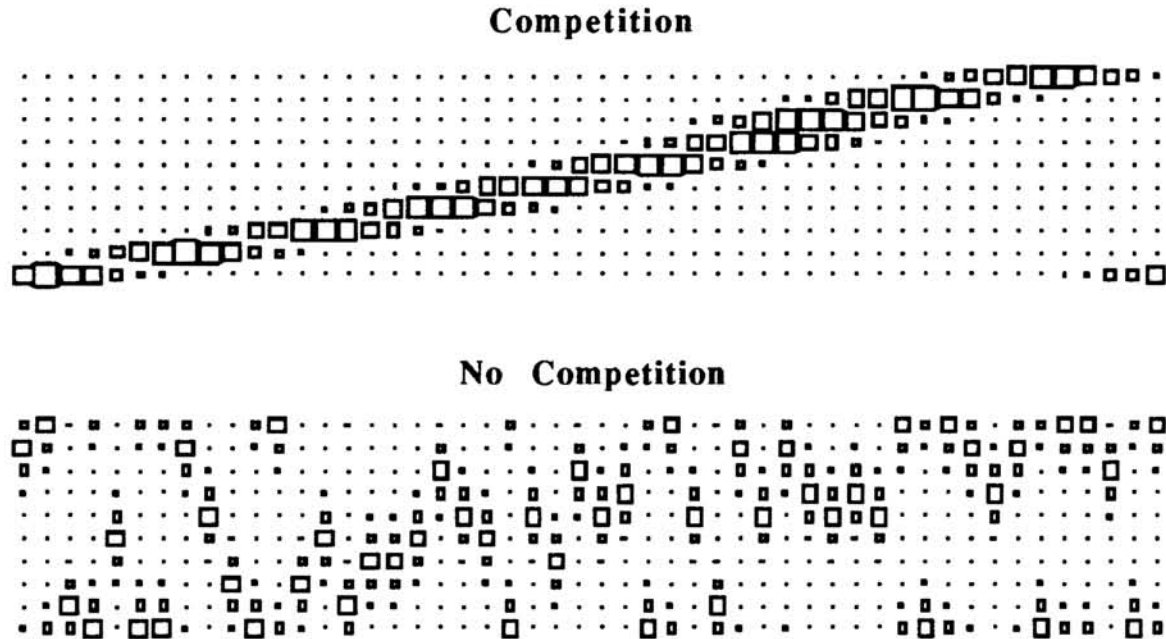

**No Competition**

Figure 3: Topographic mapping with and without competition. The vertical axis represents ten LGN cells within one section, and the horizontal axis 50 RGC cells. The size of each box indicates strength of the synapse connecting corresponding RGC and LGN cells. If the system is topographically ordered, this connection matrix should contain only connections forming a diagonal from lower left to upper right, as is found in normal development in our model (Competition). When competition is eliminated, the topographic map is coarse and non-contiguous.

## 4 PREDICTIONS

In addition to replicating current experimental findings, our model makes several interesting predictions about the outcome of novel experiments. If inputs from each eye arrive simultaneously, so that contralateral projections have no advantage in competing to innervate specific regions of the LGN, synaptic competition and retinal waves lead to a pattern of ocular dominance "patches" similar to that observed in visual cortex (Figure 2, Simultaneous). Topography is refined, but in this case a continuous map is formed between the two eyes (Figure 4) — again similar to patterns observed in visual cortex.

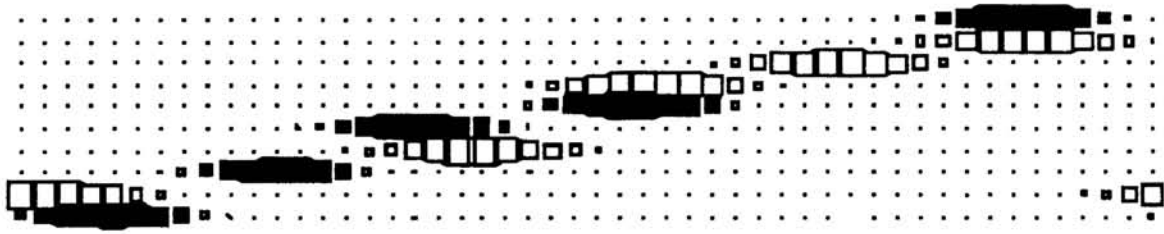

Figure 4: Topographic mapping with synchronous arrival of projections from both eyes. Light boxes represent contralateral inputs, dark boxes represent ipsilateral. Synaptic competition and retinal waves cause ocular segregation and topographic refinement, but in this case the continuous map is formed using both eyes rather than a single eye.

Our model predicts that the width of retinal waves — the distribution of activity around the moving wavefront — is an essential factor in determining both the rate of ocular segregation and topographic refinement. Wide waves, which cause many RGCs within the same eye to be active, will lead to most rapid ocular segregation as a result of competition. However, wide waves can lead to poor topography: RGCs in distant regions of the retina are just as likely to be simultaneously active as neighboring RGCs (Figure 5).

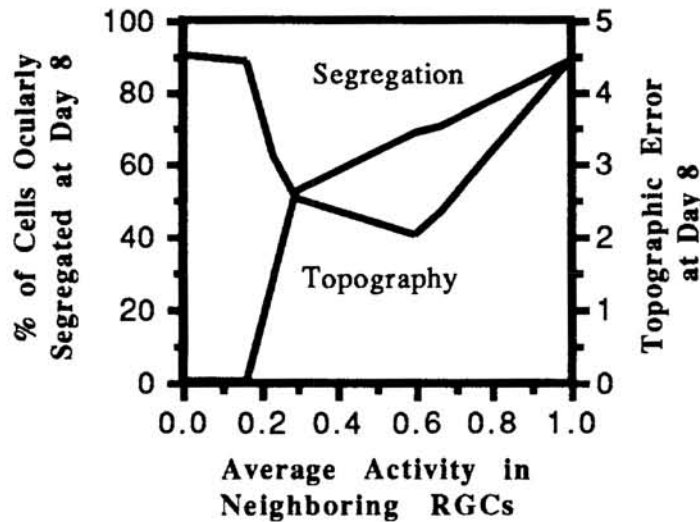

Figure 5: The width of retinal "waves" determines ocular dominance and topography in normal development in our model. Width of retinal waves is represented by the average activity in RGC cells adjacent to the Gaussian wavefront: high activity indicates wide waves. Topographic error (scale at right) represents the average distance from an RGCs target position multiplied by the strength of the synaptic connection. LGN cells are considered ocularly segregated when they receive .9 or more of their total synaptic input from one eye. Wide waves lead to rapid ocular segregation — many RGCs within the same retina are simulaneously active. An intermediate width, however, leads to lower topographic error — wide waves cause spurious correlations, while narrow waves don't provide enough information about neighboring RGCs to significantly refine topography.

# 5 SUMMARY

Our biological model differs from more developed models of cortical development in its inclusion of 1) differences in the time of arrival of RGC axons from the two eyes, 2) lack of intra-target (LGN) inhibitory connections, 3) absence of visual stimulation, and 4) inclusion of a growth rule. The model can account for the development of topography and ocular dominance layering in studies of normal and experimental-treated cats, and makes predictions concerning the role of retinal waves in both segregation and topography. These neurobiological experiments are currently underway.

### Acknowledgements

Thanks to Michael Stryker for helpful suggestions and to Steven Lisberger for his generous support of this work.

### References

Cline, H.T., Debski, E.A., & Constantine-Paton, M.. (1987) "N-methyl-D-aspartate receptor antagonist desegregates eye-specific stripes." *PNAS* **84**: 4342-4345.

Meister, M., Wong, R., Baylor, D., & Shatz, C. (1991) "Synchronous Bursts of Action Potentials in Ganglion Cells of the Developing Mammalian Retina." *Science*. **252**: 939-943.

Miller, K.D., Keller, J.B., & Stryker, M.P. (1989) "Ocular Dominance Column Development: Analysis and Simulation." *Science*. **245**: 605-615.

Shatz, C.J. (1990) "Competitive Interactions between Retinal Ganglion Cells during Prenatal Development." *J. Neurobio.* **21**(1): 197-211.

Shatz, C.J., & Stryker, M.P. (1988) "Prenatal Tetrodotoxin Infusion Blocks Segregation of Retinogeniculation Afferents." *Science*. **242**: 87-89.

Shotwell, S.L., Shatz, C.J., & Luskin, M.B. (1986) "Development of Glutamic Acid Decarboxylase Immunoreactivity in the cat's lateral geniculate nucleus." *J. Neurosci.* **6**(5) 1410-1423.

Sretavan, D.W., Shatz, C.J., & Stryker, M.P. (1988) "Modification of Retinal Ganglion Cell Morphology by Prenatal Infusion of Tetrodotoxin." *Nature*. **336**: 468-471.

Whitelaw, V.A., & Cowan, J.D. (1981) "Specificity and plasticity of retinotectal connections: a computational model." *J. Neurosci.* **1**(12) 1369-1387.

Wigstrom, H., & Gustafsson, B. (1985) "Presynaptic and postsynaptic interactions in the control of hippocampal long-term potentiation." in P.W. Landfield & S.A. Deadwyler (Eds.) *Longer-term potentiation: from biophysics to behavior* (pp. 73-107). New York: Alan R. Liss.

Wolpert, L. (1978) "Pattern Formation in Biological Development." *Sci. Amer.* **239**(4): 154-164.
